# Extracting Speaker-Specific Information with a Regularized Siamese Deep Network

**Ke Chen and Ahmad Salman**
School of Computer Science, The University of Manchester
Manchester M13 9PL, United Kingdom
{chen,salmana}@cs.manchester.ac.uk

## Abstract

Speech conveys different yet mixed information ranging from linguistic to speaker-specific components, and each of them should be exclusively used in a specific task. However, it is extremely difficult to extract a specific information component given the fact that nearly all existing acoustic representations carry all types of speech information. Thus, the use of the same representation in both speech and speaker recognition hinders a system from producing better performance due to interference of irrelevant information. In this paper, we present a deep neural architecture to extract speaker-specific information from MFCCs. As a result, a multi-objective loss function is proposed for learning speaker-specific characteristics and regularization via normalizing interference of non-speaker related information and avoiding information loss. With LDC benchmark corpora and a Chinese speech corpus, we demonstrate that a resultant speaker-specific representation is insensitive to text/languages spoken and environmental mismatches and hence outperforms MFCCs and other state-of-the-art techniques in speaker recognition. We discuss relevant issues and relate our approach to previous work.

## 1 Introduction

It is well known that speech conveys various yet mixed information where there are linguistic information, a major component, and non-verbal information such as speaker-specific and emotional components [1]. For human communication, all the information components in speech turn out to be very useful and exclusively used for different tasks. For example, one often recognizes a speaker regardless of what is spoken for speaker recognition, while it is effortless for him/her to understand what is exactly spoken by different speakers for speech recognition. In general, however, there is no effective way to automatically extract an information component of interest from speech signals so that the same representation has to be used in different speech information tasks. The interference of different yet entangled speech information components in most existing acoustic representations hinders a speech or speaker recognition system from achieving better performance [1].

For speaker-specific information extraction, two main efforts have been made so far; one is the use of data component analysis [2], e.g., PCA or ICA, and the other is the use of adaptive filtering techniques [3]. However, the aforementioned techniques either fail to associate extracted data components with speaker-specific information as such information is non-predominant over speech or obtain features overfitting to a specific corpus since it is unlikely that speaker-specific information is statically resided in fixed frequency bands. Hence, the problem is still unsolved in general [4].

Recent studies suggested that learning deep architectures (DAs) provides a new way for tackling complex AI problems [5]. In particular, representations learned by DAs greatly facilitate various recognition tasks and constantly lead to the improved performance in machine perception [6]-[9]. On the other hand, the Siamese architecture originally proposed in [10] uses supervised yet contrastive

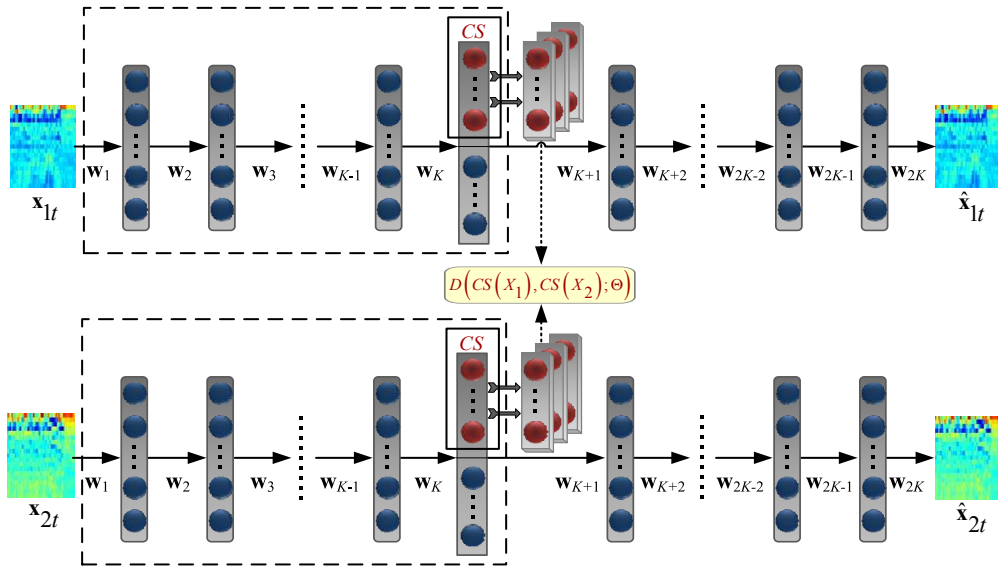

Figure 1: Regularized Siamese deep network (RSDN) architecture.

learning to explore intrinsic similarity/disimilarity underlying an unknown data space. Incorporated by DAs, the Siamese architecture has been successfully applied to face recognition [11] and dimensionality reduction [12]. Inspired by the aforementioned work, we present a *regularized Siamese deep network* (RSDN) to extract speaker-specific information from a spectral representation, *Mel Frequency Cepstral Coefficients* (MFCCs), commonly used in both speech and speaker recognition. A multi-objective loss function is proposed for learning speaker-specific characteristics, normalizing interference of non-speaker related information and avoiding information loss. Our RSDN learning adopts the famous two-phase deep learning strategy [5],[13]; i.e., greedy layer-wise unsupervised learning for initializing its component deep neural networks followed by global supervised learning based on the proposed loss function. With LDC benchmark corpora [14] and a Chinese corpus [15], we demonstrate that a generic speaker-specific representation learned by our RSDN is insensitive to text and languages spoken and, moreover, applicable to speech corpora unseen during learning. Experimental results in speaker recognition suggest that a representation learned by the RSDN outperforms MFCCs and that by the CDBN [9] that learns a generic speech representation without speaker-specific information extraction. To our best knowledge, the work presented in this paper is the first attempt on speaker-specific information extraction with deep learning.

In the reminder of this paper, Sect. 2 describes our RSDN architecture and proposes a loss function. Sect. 3 presents a two-phase learning algorithm to train the RSDN. Sect. 4 reports our experimental methodology and results. The last section discusses relevant issues and relates our approach to previous work in deep learning.

## 2 Model Description

In this section, we first describe our RSDN architecture and then propose a multi-objective loss function used to train the RSDN for learning speaker-specific characteristics.

### 2.1 Architecture

As illustrated in Figure 1, our RSDN architecture consists of two subnets, and each subnet is a fully connected multi-layered perceptron of *2K+1* layers, i.e., an input layer, *2K-1* hidden layers and a visible layer at the top. If we stipulate that layer *0* is input layer, there are the same number of neurons in layers *k* and *2K-k* for $k = 0, 1, \cdots, K$. In particular, the *K*th hidden layer is used as *code layer*, and neurons in this layer are further divided into two subsets. As depicted in Figure 1, those neurons in the box named $\mathcal{CS}$ and colored in red constitute one subset for encoding speaker-specific information and all remaining neurons in the code layer form the other subset expected to

accommodate non-speaker related information. The input to each subnet is an MFCC representation of a frame after a short-term analysis that a speech segment is divided into a number of frames and the MFCC representation is achieved for each frame. As depicted in Figure 1, $\boldsymbol{x}_{it}$ is the MFCC feature vector of frame $t$ in $X_i$, input to subnet $i$ ($i=1,2$), where $X_i = \{\boldsymbol{x}_{it}\}_{t=1}^{T_B}$ collectively denotes MFCC feature vectors for a speech segment of $T_B$ frames.

During learning, two identical subsets are coupled at their coding layers via neurons in $\mathcal{CS}$ with an incompatibility measure defined on two speech segments of equal length, $X_1$ and $X_2$, input to two subnets, which will be presented in 2.2. After learning, we achieve two identical subnets and hence can use either of them to produce a new representation for a speech frame. For input $\boldsymbol{x}$ to a subnet, only the bottom $K$ layers of the subnet are used and the output of neurons in $\mathcal{CS}$ at the code layer or layer $K$, denoted by $\mathcal{CS}(\boldsymbol{x})$, is its new representation, as illustrated by the dash box in Figure 1.

## 2.2 Loss Function

Let $\mathcal{CS}(\boldsymbol{x}_{it})$ be the output of all neurons in $\mathcal{CS}$ of subnet $i$ ($i=1,2$) for input $\boldsymbol{x}_{it} \in X_i$ and $\mathcal{CS}(X_i) = \{\mathcal{CS}(\boldsymbol{x}_{it})\}_{t=1}^{T_B}$, which pools output of neurons in $\mathcal{CS}$ for $T_B$ frames in $X_i$, as illustrated in Figure 1. As statistics of speech signals is more likely to capture speaker-specific information [5], we define the incompatibility measure based on the 1st- and 2nd-order statistics of a new representation to be learned as

$$D[\mathcal{CS}(X_1), \mathcal{CS}(X_2); \Theta] = ||\boldsymbol{\mu}^{(1)} - \boldsymbol{\mu}^{(2)}||_2^2 + ||\Sigma^{(1)} - \Sigma^{(2)}||_F^2, \qquad (1)$$

where

$$\boldsymbol{\mu}^{(i)} = \frac{1}{T_B}\sum_{t=1}^{T_B} \mathcal{CS}(\boldsymbol{x}_{it}), \;\; \Sigma^{(i)} = \frac{1}{T_B - 1}\sum_{t=1}^{T_B}[\mathcal{CS}(\boldsymbol{x}_{it}) - \boldsymbol{\mu}^{(i)}][\mathcal{CS}(\boldsymbol{x}_{it}) - \boldsymbol{\mu}^{(i)}]^T, \;\; i = 1, 2.$$

In Eq. (1), $||\cdot||_2$ and $||\cdot||_F$ are the $\mathcal{L}_2$ norm and the Frobenius norm, respectively. $\Theta$ is a collective notation of all connection weights and biases in the RSDN. Intuitively, two speech segments belonging to different speakers lead to different statistics and hence their incompatibility score measured by (1) should be large after learning. Otherwise their score is expected to be small.

For a corpus of multiple speakers, we can construct a training set so that an example be in the form: $(X_1, X_2; \mathcal{I})$ where $\mathcal{I}$ is the label defined as $\mathcal{I} = 1$ if two speech segments, $X_1$ and $X_2$, are spoken by the same speaker or $\mathcal{I} = 0$ otherwise. Using such training examples, we apply the energy-based model principle [16] to define a loss function as

$$L(X_1, X_2; \Theta) = \alpha[L_R(X_1; \Theta) + L_R(X_2; \Theta)] + (1 - \alpha)L_D(X_1, X_2; \Theta), \qquad (2)$$

where

$$L_R(X_i; \Theta) = \frac{1}{T_B}\sum_{t=1}^{T_B} ||\boldsymbol{x}_{it} - \hat{\boldsymbol{x}}_{it}||_2^2 \; (i=1,2), \;\; L_D(X_1, X_2; \Theta) = \mathcal{I}D + (1 - \mathcal{I})(e^{-\frac{D_m}{\lambda_m}} + e^{-\frac{D_S}{\lambda_S}}).$$

Here $D_m = ||\boldsymbol{\mu}^{(1)} - \boldsymbol{\mu}^{(2)}||_2^2$ and $D_S = ||\Sigma^{(1)} - \Sigma^{(2)}||_F^2$. $\lambda_m$ and $\lambda_S$ are the tolerance bounds of incompatibility scores in terms of $D_m$ and $D_S$, which can be estimated from a training set. In $L_D(X_1, X_2; \Theta)$, we drop explicit parameters of $D[\mathcal{CS}(X_1), \mathcal{CS}(X_2); \Theta]$ to simplify presentation.

Eq. (2) defines a multi-objective loss function where $\alpha$ ($0 < \alpha < 1$) is a parameter used to trade-off between two objectives $L_R(X_i; \Theta)$ and $L_D(X_1, X_2; \Theta)$. The motivation for two objectives are as follows. By nature, both speaker-specific and non-speaker related information components are entangled over speech [1],[5]. When we tend to extract speaker-specific information, the interference of non-speaker related information is inevitable and appears in various forms. $L_D(X_1, X_2; \Theta)$ measures errors responsible for wrong speaker-specific statistics on a representation learned by a Siamese DA in different situations. However, using $L_D(X_1, X_2; \Theta)$ only to train a Siamese DA cannot cope with enormous variations of non-speaker related information, in particular, linguistic information (a predominant information component in speech), which often leads to overfitting to a training corpus according to our observations. As a result, we use $L_R(X_i; \Theta)$ to measure reconstruction errors to monitor information loss during speaker-specific information extraction. By minimizing reconstruction errors in two subnets, the code layer leads to a speaker-specific representation with the output of neurons in $\mathcal{CS}$ while the remaining neurons are used to regularize various interference by capturing some invariant properties underlying them for good generalization.

In summary, we anticipate that minimizing the multi-objective loss function defined in Eq. (2) will enable our RSDN to extract speaker-specific information by encoding it through a generic speaker-specific representation.

# 3 Learning Algorithm

In this section, we apply the two-phase deep learning strategy [5],[13] to derive our learning algorithm, i.e., pre-training for initializing subnets and discriminative learning for learning a speaker-specific representation.

We first present the notation system used in our algorithm. Let $h_{kj}(\boldsymbol{x}_{it})$ denote the output of the $j$th neuron in layer $k$ for $k=0,1,\cdots,K,\cdots,2K$. $\boldsymbol{h}_k(\boldsymbol{x}_{it}) = \left(h_{kj}(\boldsymbol{x}_{it})\right)_{j=1}^{|\boldsymbol{h}_k|}$ is a collective notation of the output of all neurons in layer $k$ of subnet $i$ ($i=1,2$) where $|\boldsymbol{h}_k|$ is the number of neurons in layer $k$. By this notation, $k=0$ refers to the input layer with $\boldsymbol{h}_0(\boldsymbol{x}_{it}) = \boldsymbol{x}_{it}$, and $k=2K$ refers to the top layer producing the reconstruction $\hat{\boldsymbol{x}}_{it}$. In the coding layer, i.e., layer $K$, $\mathcal{CS}(\boldsymbol{x}_{it}) = \left(h_{Kj}(\boldsymbol{x}_{it})\right)_{j=1}^{|\mathcal{CS}|}$ is a simplified notation for output of neurons in $\mathcal{CS}$. Let $W_k^{(i)}$ and $\boldsymbol{b}_k^{(i)}$ denote the connection weight matrix between layers $k-1$ and $k$ and the bias vector of layer $k$ in subnet $i$ ($i=1,2$), respectively, for $k=1,\cdots,2K$. Then output of layer $k$ is $\boldsymbol{h}_k(\boldsymbol{x}_{it}) = \sigma[\boldsymbol{u}_k(\boldsymbol{x}_{it})]$ for $k=1,\cdots,2K\text{-}1$, where $\boldsymbol{u}_k(\boldsymbol{x}_{it}) = W_k^{(i)}\boldsymbol{h}_{k-1}(\boldsymbol{x}_{it}) + \boldsymbol{b}_k^{(i)}$ and $\sigma(\boldsymbol{z}) = \left((1 + e^{-z_j})^{-1}\right)_{j=1}^{|\boldsymbol{z}|}$. Note that we use the linear transfer function in the top layer, i.e., layer $2K$, to reconstruct the original input.

## 3.1 Pre-training

For pre-training, we employ the denoising autoencoder [17] as a building block to initialize biases and connection weight matrices of a subnet. A denoising autoencoder is a three-layered perceptron where the input, $\tilde{\boldsymbol{x}}$, is a distorted version of the target output, $\boldsymbol{x}$. For a training example, $(\tilde{\boldsymbol{x}}, \boldsymbol{x})$, the output of the autoencoder is a restored version, $\hat{\boldsymbol{x}}$. Since MFCCs fed to the first hidden layer and its intermediate representation input to all other hidden layers are of continuous value, we always distort input, $\boldsymbol{x}$, by adding Gaussian noise to form a distorted version, $\tilde{\boldsymbol{x}}$. The restoration learning is done by minimizing the MSE loss between $\boldsymbol{x}$ and $\hat{\boldsymbol{x}}$ with respect to the weight matrix and biases. We apply the stochastic back-propagation (SBP) algorithm to train denoising autoencoders, and the greedy layer-wise learning procedure [5],[13] leads to initial weight matrices for the first $K$ hidden layers, as depicted in a dash box in Figure 1, i.e., $W_1,\cdots,W_K$ of a subnet. Then, we set $W_{K+k} = W_{K-k+1}^T$ for $k=1,\cdots,K$ to initialize $W_{K+1},\cdots,W_{2K}$ of the subnet. Finally, the second subnet is created by simply duplicating the pre-trained one.

## 3.2 Discriminative Learning

For discriminative learning, we minimizing the loss function in Eq. (2) based on pre-trained subnets for speaker-specific information extraction. Given our loss function is defined on statistics of $T_B$ frames in a speech segment, we cannot update parameters until we have $T_B$ output of neurons in $\mathcal{CS}$ at the code layer. Fortunately, the SBP algorithm perfectly meets our requirement; In the SBP algorithm, we always set the batch size to the number of frames in a speech segment. To simplify the presentation, we shall drop explicit parameters in our derivation if doing so causes no ambiguities.

In terms of the reconstruction loss, $L_R(X_i; \Theta)$, we have the following gradients. For layer $k = 2K$,

$$\frac{\partial L_R}{\partial \boldsymbol{u}_{2K}(\boldsymbol{x}_{it})} = 2(\hat{\boldsymbol{x}}_{it} - \boldsymbol{x}_{it}), \ \ i=1,2. \tag{3}$$

For all hidden layers, $k=2K\text{-}1,\cdots,1$, applying the chain rule and (3) leads to

$$\frac{\partial L_R}{\partial \boldsymbol{u}_k(\boldsymbol{x}_{it})} = \left(\frac{\partial L_R}{\partial h_{kj}(\boldsymbol{x}_{it})}h_{kj}(\boldsymbol{x}_{it})[1-h_{kj}(\boldsymbol{x}_{it})]\right)_{j=1}^{|\boldsymbol{h}_k|}, \ \ \frac{\partial L_R}{\partial \boldsymbol{h}_k(\boldsymbol{x}_{it})} = \left[W_{k+1}^{(i)}\right]^T \frac{\partial L_R}{\partial \boldsymbol{u}_{k+1}(\boldsymbol{x}_{it})}. \tag{4}$$

As the contrastive loss, $L_D(X_1, X_2; \Theta)$, defined on neurons in $\mathcal{CS}$ at code layers of two subnets, its gradients are determined only by parameters related to $K$ hidden layers in two subnets, as depicted by dash boxes in Figure 1. For layer $k=K$ and subnet $i=1, 2$, after a derivation (see the appendix for details), we obtain

$$\frac{\partial L_D}{\partial \boldsymbol{u}_K(\boldsymbol{x}_{it})} = \left(([\mathcal{I} - \lambda_m^{-1}(1 - \mathcal{I})e^{-\frac{D_m}{\lambda_m}}]\psi_j(\boldsymbol{x}_{it}))_{j=1}^{|\mathcal{CS}|}, (0)_{j=|\mathcal{CS}|+1}^{|\boldsymbol{h}_K|}\right) +$$
$$\left(([\mathcal{I} - \lambda_S^{-1}(1 - \mathcal{I})e^{-\frac{D_S}{\lambda_S}}]\xi_j(\boldsymbol{x}_{it}))_{j=1}^{|\mathcal{CS}|}, (0)_{j=|\mathcal{CS}|+1}^{|\boldsymbol{h}_K|}\right). \tag{5}$$

Here, $\psi_j(\boldsymbol{x}_{it}) = p_j^{(i)} \big(\mathcal{CS}(\boldsymbol{x}_{it})\big)_j \big[1 - \big(\mathcal{CS}(\boldsymbol{x}_{it})\big)_j\big]$ and $\xi_j(\boldsymbol{x}_{it}) = q_j(\boldsymbol{x}_{it}) \big(\mathcal{CS}(\boldsymbol{x}_{it})\big)_j \big[1 - \big(\mathcal{CS}(\boldsymbol{x}_{it})\big)_j\big]$, where $\boldsymbol{p}^{(i)} = \frac{2}{T_B}\mathrm{sign}(1.5-i)(\boldsymbol{\mu}^{(1)} - \boldsymbol{\mu}^{(2)})$, $\boldsymbol{q}(\boldsymbol{x}_{it}) = \frac{4}{T_B-1}\mathrm{sign}(1.5-i)(\Sigma^{(1)} - \Sigma^{(2)})[\mathcal{CS}(\boldsymbol{x}_{it}) - \boldsymbol{\mu}^{(i)}]$ and $\big(\mathcal{CS}(\boldsymbol{x}_{it})\big)_j$ is output of the $j$th neuron in $\mathcal{CS}$ for input $\boldsymbol{x}_{it}$. For layers $k=K\text{-}1, \cdots, 1$, we have

$$\frac{\partial L_D}{\partial \boldsymbol{u}_k(\boldsymbol{x}_{it})} = \left(\frac{\partial L_D}{\partial h_{kj}(\boldsymbol{x}_{it})}h_{kj}(\boldsymbol{x}_{it})[1-h_{kj}(\boldsymbol{x}_{it})]\right)_{j=1}^{|\boldsymbol{h}_k|}, \quad \frac{\partial L_D}{\partial \boldsymbol{h}_k(\boldsymbol{x}_{it})} = \big[W_{k+1}^{(i)}\big]^T \frac{\partial L_R}{\partial \boldsymbol{u}_{k+1}(\boldsymbol{x}_{it})}. \quad (6)$$

Given a training example, $\big(\{\boldsymbol{x}_{1t}\}_{t=1}^{T_B}, \{\boldsymbol{x}_{2t}\}_{t=1}^{T_B}; \mathcal{I}\big)$, we use gradients achieved from Eqs. (3)-(6) to update all the parameters in the RSDN. For layers $k=K\text{+}1, \cdots, 2K$, their parameters are updated by

$$W_k^{(i)} \leftarrow W_k^{(i)} - \frac{\epsilon\alpha}{T_B}\sum_{t=1}^{T_B}\sum_{r=1}^{2}\frac{\partial L_R}{\partial \boldsymbol{u}_k(\boldsymbol{x}_{rt})}[\boldsymbol{h}_{k-1}(\boldsymbol{x}_{rt})]^T, \quad \boldsymbol{b}_k^{(i)} \leftarrow \boldsymbol{b}_k^{(i)} - \frac{\epsilon\alpha}{T_B}\sum_{t=1}^{T_B}\sum_{r=1}^{2}\frac{\partial L_R}{\partial \boldsymbol{u}_k(\boldsymbol{x}_{rt})}. \quad (7)$$

For layers $k=1, \cdots, K$, their weight matrices and biases are updated with

$$W_k^{(i)} \leftarrow W_k^{(i)} - \frac{\epsilon}{T_B}\sum_{t=1}^{T_B}\sum_{r=1}^{2}\left(\alpha\frac{\partial L_R}{\partial \boldsymbol{u}_k(\boldsymbol{x}_{rt})} + (1-\alpha)\frac{\partial L_D}{\partial \boldsymbol{u}_k(\boldsymbol{x}_{rt})}\right)[\boldsymbol{h}_{k-1}(\boldsymbol{x}_{rt})]^T, \quad (8\text{a})$$

$$\boldsymbol{b}_k^{(i)} \leftarrow \boldsymbol{b}_k^{(i)} - \frac{\epsilon}{T_B}\sum_{t=1}^{T_B}\sum_{r=1}^{2}\left(\alpha\frac{\partial L_R}{\partial \boldsymbol{u}_k(\boldsymbol{x}_{rt})} + (1-\alpha)\frac{\partial L_D}{\partial \boldsymbol{u}_k(\boldsymbol{x}_{rt})}\right). \quad (8\text{b})$$

In Eqs. (7) and (8), $\epsilon$ is a learning rate. Here we emphasize that using sum of gradients caused by two subnets in update rules guarantees that two subsets are always kept identical during learning.

## 4 Experiment

In this section, we describe our experimental methodology and report experiments results in visualization of vowel distributions, speaker comparison and speaker segmentation.

We employ two LDC benchmark corpora [14], KING and TIMIT, and a Chinese speech corpus [15], CHN, in our experiments. KING, including wide-band and narrow-band sets, consists of 51 speakers whose utterances were recorded in 10 sessions. By convention, its narrow-band set is called NKING while KING itself is often referred to its wide-band set. There are 630 speakers in TIMIT and 59 speakers in CHN of three sessions, respectively. All corpora were collected especially for evaluating a speaker recognition system. The same feature extraction procedure is applied to all three corpora; i.e., after a short-term analysis suggested in [18], including silence removal with an energy-based method, pre-emphasis with the filter $H(z) = 1 - 0.95z^{-1}$ as well as Hamming windowing with the size of 20 ms and 10 ms shift, we extract 19-order MFCCs [1] for each frame.

For the RSDN learning, we use utterances of all 49 speakers recorded in sessions 1 and 2 in KING. Furthermore, we distort all the utterances by the additive white noise channel with SNR of 10dB and the Rayleigh fading channel with 5 Hz Doppler shift [19] to simulate channel effects. Thus our training set consists of clean utterances and their corrupted versions. We randomly divide all utterances into speech segments of a length $T_B$ ($1\,\mathrm{sec} \le T_B \le 2\,\mathrm{sec}$) and then exhaustively combine them to form training examples as described in Sect. 2.2. With a validation set of all the utterances recorded in session 3 in KING, we select a structure of $K=4$ (100, 100, 100 and 200 neurons in layers 1-4 and $|\mathcal{CS}|=100$ in the code layer or layer 4) from candidate models of $2<K<5$ and 50-1000 neurons in a hidden layer. Parameters used in our learning are as follows: Gaussian noise of $N(0, 0.1\sigma)$ used in denoising autoencoder, $\alpha=0.2$, $\lambda_m=100$ and $\lambda_S=2.5$ in the loss function defined in Eq. (2), and learning rates $\epsilon=0.01$ and 0.001 for pre-training and discriminative learning. After learning, the RSDN is used to yield a 100-dimensional representation, $\mathcal{CS}$, from 19-order MFCCs.

For any speaker recognition tasks, speaker modeling (SM) is inevitable. In our experiments, we use the 1st- and 2nd-order statistics of a speech segment based on a representation, $\mathcal{SM} = \{\boldsymbol{\mu}, \Sigma\}$, for SM. Furthermore, we employ a speaker distance metric: $d(\mathcal{SM}_1, \mathcal{SM}_2) = \mathrm{tr}[(\Sigma_1^{-1} + \Sigma_2^{-1})(\boldsymbol{\mu}_1 - \boldsymbol{\mu}_2)(\boldsymbol{\mu}_1 - \boldsymbol{\mu}_2)^T]$, where $\mathcal{SM}_i = \{\boldsymbol{\mu}_i, \Sigma_i\}$ ($i=1,2$) are two speaker models ($\mathcal{SM}$s). This distance metric is derived from the divergence metric for two normal distributions [20] by dropping the term concerning only covariance matrices based on our observation that covariance matrices often vary considerably for short segments and the original divergence metric often leads to poor performance for various representations including MFCCs and ours. In contrast, the one defined above is stable irrespective of utterance lengths and results in good performance for different representations.

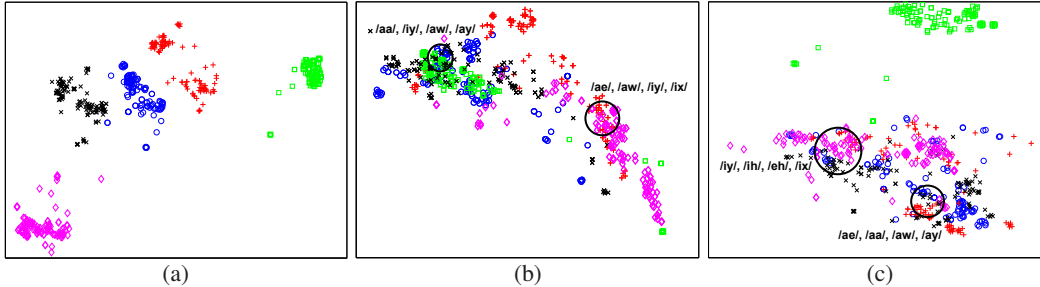

Figure 2: Visualization of all 20 vowels. (a) $\mathcal{CS}$ representation. (b) $\overline{\mathcal{CS}}$ representation. (c) MFCCs.

## 4.1 Visualization

Vowels have been recognized to be a main carrier of speaker-specific information [1],[4],[18],[20]. TIMIT [14] provides phonetic transcription of all 10 utterances containing all 20 vowels in English for every speaker. As all the vowels may appear in 10 different utterances, up to 200 vowel segments in length of 0.1-0.5 sec are available for a speaker, which enables us to investigate vowel distributions in a representation space for different speakers. Here, we merely visualize mean feature vectors of up to 200 segments for a speaker in terms of a specific representation with the t-SNE method [21], which is likely to reflect intrinsic manifolds, by projecting them onto a two-dimensional plane.

In the code layer of our RSDN, output of neurons 1-100 forms a speaker-specific representation, $\mathcal{CS}$, and that of remaining 100 neurons becomes a non-speaker related representation, dubbed $\overline{\mathcal{CS}}$. For a noticeable effect, we randomly choose only five speakers (four females and one male) and visualize their vowel distributions in Figure 2 in terms of $\mathcal{CS}$, $\overline{\mathcal{CS}}$ and MFCC representations, respectively, where a maker/color corresponds to a speaker. It is evident from Figure 2(a) that, by using the $\mathcal{CS}$ representation, most vowels spoken by a speaker are tightly grouped together while vowels spoken by different speakers are well separated. For the $\overline{\mathcal{CS}}$ representation, close inspection on Figure 2(b) reveals that the same vowels spoken by different speakers are, to a great extent, co-located. Moreover, most of phonetically correlated vowels, as circled and labeled, are closely located in dense regions independent of speakers and genders. For comparison, we also visualize the same by using their original MFCCs in Figure 2(c) and observe that most of phonetically correlated vowels are also co-located, as circled and labeled, whilst others scatter across the plane and their positions are determined mainly by vowels but affected by speakers. In particular, most of vowels spoken by the male, marked by □ and colored by green, are grouped tightly but isolated from those by all females. Thus, visualization in Figure 2 demonstrates how our RSDN learning works and could lend an evidence to justification on why MFCCs can be used in both speech and speaker recognition [1].

## 4.2 Speaker Comparison

Speaker comparison (SC) is an essential process involved in any speaker recognition tasks by comparing two speaker models to collect evidence for decision-making, which provides a direct way to evaluate representations/speaker modeling without addressing decision-making issues [22]. In our SC experiments, we employ NKING [14], a narrow-band corpus, of many variabilities. During data collection, there was a "great divide" between sessions 1-5 and 6-10; both recording device and environments changed, which alters spectral features of 26 speakers and leads to 10dB SNR reduction on average. As suggested in [18], we conduct two experiments: *within-divide* where $\mathcal{SM}$s built on utterances in session 1 are compared to $\mathcal{SM}$s on those in sessions 2-5 and *cross-divide* where $\mathcal{SM}$s built on utterances in session 1 are compared with those in sessions 6-10. As short utterances poses a greater challenge for speaker recognition [4],[18],[20], utterances are partitioned into short segments of a certain length and $\mathcal{SM}$s built on segments of the same length are always used for SC. For a thorough evaluation, we apply the SM technique in question to our representation, MFCCs, and a representation (i.e., the better one of those yielded by two layers) learned by the CDBN [9] on all 10 sessions in NKING, and name them *SM-RSDN, SM-MFCC* and *SM-CDBN* hereinafter. In addition, we also compare them to GMMs trained on MFCCs (*GMM-MFCC*), a state-of-the-art SM technique that provides the baseline performance [4],[20], where for each speaker a GMM-based $\mathcal{SM}$ consisting of 32 Gaussian components is trained on his/her utterances of 60 sec in sessions 1-2 with the EM algorithm [18]. For the CDBN learning [9] and the GMM training [18], we strictly follow their suggested parameter settings in our experiments (see [9],[18] for details).

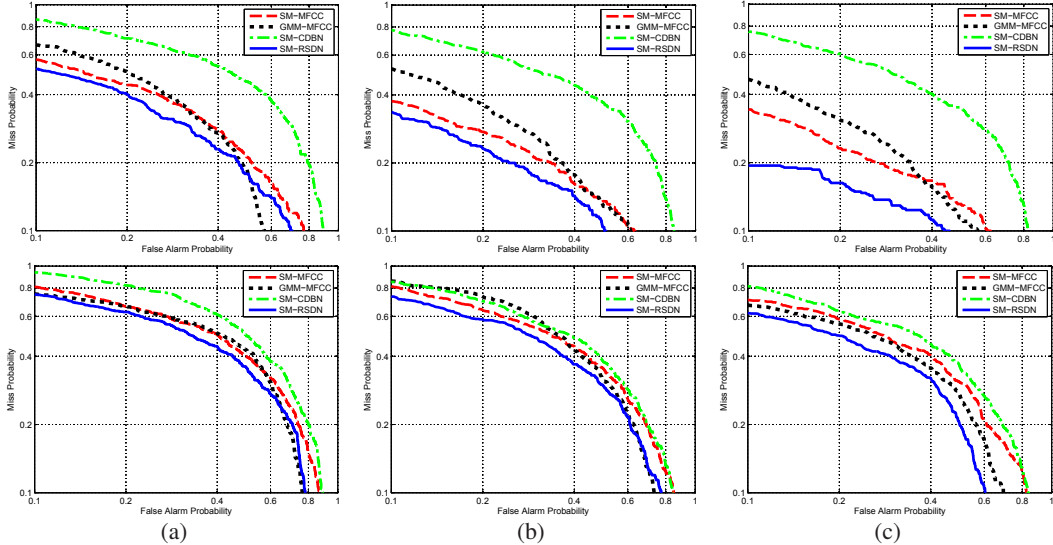

Figure 3: Performance of speaker comparison (DET) in the within-divide (upper row) and the cross-divide (lower row) experiments for different segment lengths. (a) 1 sec. (b) 3 sec. (d) 5 sec.

Table 1: Performance (mean±std)% of speaker segmentation on TIMIT and CHN audio streams.

| Index | TIMIT Audio Stream | | | CHN Audio Stream | | |
|---|---|---|---|---|---|---|
| | BIC-MFCC | Dist-MFCC | Dist-RSDN | BIC-MFCC | Dist-MFCC | Dist-RSDN |
| FAR | 26±09 | 22±11 | **18±11** | 46±04 | 27±11 | **24±11** |
| MDR | 26±14 | 22±12 | **18±10** | 46±10 | 27±17 | **24±17** |
| $F_1$ | 67±12 | 74±11 | **79±09** | 44±08 | 68±17 | **72±17** |

We use *Detection Error Trade-off* (DET) curves as the performance index in SC. From Figure 3, it is evident that SM-RSDN outperforms SM-MFCC, SM-CDBN and GMM-MFCC, a baseline system trained on much longer utterances, as it always yields a smaller operating region, i.e., all possible errors, in all the settings. In contrast, SM-MFCC performs better in within-divide settings while SM-CDBN is always inferior to the baseline system. Relevant issues will be discussed later on.

### 4.3 Speaker Segmentation

Speaker segmentation (SS) is a task of detecting speaker change points in an audio stream to split it into acoustically homogeneous segments so that every segment contains only one speaker [23]. Following the same protocol used in previous work [23], we utilize utterances in TIMIT and CHN corpora to simulate audio conversations. As a result, we randomly select 250 speakers from TIMIT to create 25 audio streams where the duration of speakers ranges from 1.6 to 7.0 sec and 50 speakers from CHN to create 15 audio streams where the duration of speakers is from 3.0 to 8.3 sec. In the absence of prior knowledge, the distance-based and the BIC techniques are two main approaches to SS [23]. In our simulations, we apply the distance-based method [23] to our representation and MFCCs, dubbed *Dist-RSDN* and *Dist-MFCC*, where the same parameters, including sliding window of 1.5 sec and tolerance level of 0.5 sec, are used. In addition, we also apply the BIC method [23] to MFCCs (*BIC-MFCC*). Note that the BIC method is inapplicable to our representation since it uses only covariance information but the high dimensionality of our representation and the use of a small sliding window in the BIC result in unstable performance, as pointed out early in this section.

For evaluation, we use three common indexes [23], i.e., *False Alarm Rate* (FAR), *Miss Detection Rate* (MDR) and $F_1$ measure defined based on both precision and recall rates. Moreover, we only report results as FAR equals MDR to avoid addressing decision-making issues [23]. Table 1 tabulates SS performance where, as boldfaced, results by our representation are superior to those by MFCCs regardless of SS methods and corpora for creating audio streams used in our simulations.

In summary, visualization of vowels and results in SC and SS suggest that our RSDN successfully extracts speaker-specific information; its resultant representation can be generalized to unseen corpora during learning and is insensitive to text and languages spoken and environmental changes.

# 5 Discussion

As pointed out earlier, speech carries different yet mixed information and speaker-specific information is minor in comparison to predominant linguistic information. Our empirical studies suggest that our success in extracting speaker-specific information is attributed to both unsupervised pre-training and supervised discriminative learning with a contrastive loss. In particular, the use of data regularization in discriminative learning and distorted data in two learning phases plays a critical role in capturing intrinsic speaker-specific characteristics and variations caused by miscellaneous mismatches. Our results not reported here, due to limited space, indicate that without the pre-training in Sect. 3.1, a randomly initialized RSDN leads to unstable performance often considerably worse than that of using the pre-training in general. Without discriminative learning, a DA working on unsupervised learning only, e.g., the CDBN [9], tends to yield a new representation that redistributes different information but does not highlight minor speaker-specific information given the fact that the CDBN trained on all 10 sessions in NKING leads to a representation that fails to yield satisfactory SC performance on the same corpus but works well for various audio classification tasks [9]. If we do not use the regularization term, $L_R(X_i; \Theta)$, in the loss function in Eq. (2), our RSDN is boiled down to a standard Siamese architecture [10]. Our results not reported here show that such an architecture learns a representation often overfitting to the training corpus due to interference of predominant non-speaker related information, which is not a problem in predominant information extraction. The previous work in face recognition [11] could lend an evidence to support our argument where a Siamese DA without regularization successfully captures predominant identity characteristics from facial images as, we believe, facial expression and other non-identity information are minor in this situation. While the use of distorted data in pre-training is in the same spirit of self-taught learning [24], our results including those not reported here reveal that the use of distorted data in pre-training but not in discriminative learning yields results worse than the baseline performance in the cross-divide SC experiment. Hence, sufficient training data reflecting mismatches are also required in discriminative learning for speaker-specific information extraction.

Our RSDN architecture resembles the one proposed in [12] for dimensionality reduction of handwritten digits via learning a nonlinear embedding. However, ours distinguishes from theirs in the use of different building blocks in our DAs, loss functions and motivations. The DA in [12] uses the RBM [13] as a building block to construct a deep belief subnet in their Siamese DA and the NCA [25] as their contrastive loss function to minimize the intra-class variability. However, the NCA does not meet our requirements as there are so many examples in one class. Instead we propose a contrastive loss to minimize both intra- and inter-class variabilities simultaneously. On the other hand, intrinsic topological structures of a handwritten digit convey predominant information given the fact that without using the NCA loss a deep belief autoencoder already yields a good representation [7],[12],[13],[26]. Thus, the use of the NCA in [12] simply reinforces the topological invariance by minimizing other variabilities with a small amount of labeled data [12]. In our work, however, speaker-specific information is non-predominant in speech and hence a large amount of labeled data reflecting miscellaneous variabilities are required during discriminative learning despite the pre-training. Finally, our code layer yields an overcomplete representation to facilitate non-predominant information extraction. In contrast, a parsimonious representation seems more suitable for extracting predominant information since dimensionality reduction is likely to discover "principal" components that often associate with predominant information, as are evident in [11],[12].

To conclude, we propose a deep neural architecture for speaker-specific information extraction and demonstrate that its resultant speaker-specific representation outperforms the state-of-the-art techniques. It should also be stated that our work presented here is limited to speech corpora available at present. In our ongoing work, we are employing richer training data towards learning a universal speaker-specific representation. In a broader sense, our work presented in this paper suggests that speech *information component analysis* (ICA) becomes critical in various speech information processing tasks; the use of proper speech ICA techniques would result in task-specific speech representations to improve their performance radically. Our work demonstrates that speech ICA is feasible via learning. Moreover, deep learning could be a promising methodology for speech ICA.

**Acknowledgments**

Authors would like to thank H. Lee for providing their CDBN code [9] and L. Wang for offering their SIAT Chinese speech corpus [15] to us; both of which were used in our experiments.

# References

[1] Huang, X., Acero, A. & Hon, H. (2001) *Spoken Language Processing.* New York: Prentice Hall.

[2] Jang, G., Lee, T. & Oh, Y. (2001) Learning statistically efficient feature for speaker recognition. *Proc. ICASSP*, pp. I427-I440, IEEE Press.

[3] Mammone, R., Zhang, X. & Ramachandran, R. (1996) Robust speaker recognition: a feature-based approach. *IEEE Signal Processing Magazine*, **13**(1): 58-71.

[4] Reynold, D. & Campbell, W. (2008) Text-independent speaker recognition. In J. Benesty, M. Sondhi and Y. Huang (Eds.), *Handbook of Speech Processing*, pp. 763-781, Berlin: Springer.

[5] Bengio, Y. (2009) Learning deep architectures for AI. *Foundation and Trends in Machine Learning* **2**(1): 1-127.

[6] Hinton, G. (2007) Learning multiple layers of representation. *Trends in Cognitive Science* **11**(10): 428-434.

[7] Larochelle, H., Bengio, Y., Louradour, J. & Lamblin, P. (2009) Exploring strategies for training deep neural networks. *Journal of Machine Learning Research* **10**(1): pp. 1-40.

[8] Boureau, Y., Bach, F., LeCun, Y. & Ponce, J. (2010) Learning mid-level features for recognition. *Proc. CVPR*, IEEE Press.

[9] Lee, H., Largman, Y., Pham, P. & Ng, A. (2009) Unsupervised feature learning for audio classification using convolutional deep belief networks. In *Advances in Neural Information Processing Systems 22*, Cambridge, MA: MIT Press.

[10] Bromley, J., Guyon, I., LeCun, Y., Sackinger, E. & Shah, R. (1994) Signature verification using a Siamese time delay neural network. In *Advances in Neural Information Processing Systems 5*, Morgan Kaufmann.

[11] Chopra, S., Hadsell, R. & LeCun, Y. (2005) Learning a similarity metric discriminatively, with application to face verification. In *Proc. CVPR*, IEEE Press.

[12] Salakhutdinov, R. & Hinton, G. (2007) Learning a non-linear embedding by preserving class neighborhood structure. In *Proc. AISTATS*, Cambridge, MA: MIT Press.

[13] Hinton, G., Osindero, S. & Teh, Y. (2006) A fast learning algorithm for deep belief nets. *Neural Computation* **18**(7): 1527-1554.

[14] Linguistic Data Consortium (LDC). [online] www.ldc.upenn.edu

[15] Wang, L. (2008) A Chinese speech corpus for speaker recognition. *Tech. Report*, SIAT-CAS, China.

[16] LeCun, Y., Chopra, S. Hadsell, R., Ranzato, M. & Huang, F. (2007) Energy-based models. In *Predicting Structured Outputs*, pp. 191-246, Cambridge, MA: MIT Press.

[17] Vincent, P., Bengio, Y. & Manzagol, P. (2008) Extracting and composing robust features with denoising autoencoders. *Proc. ICML*, pp. 1096-1102, ACM Press.

[18] Reynolds, D. (1995) Speaker Identification and verification using Gaussian mixture speaker models. *Speech Communication* **17**(1): 91-108.

[19] Proakis, J. (2001) *Digital Communications (4th Edition)*. New York: McGraw-Hill.

[20] Campbell, J. (1997) Speaker recognition: A tutorial. *Proceedings of The IEEE* **85**(10): 1437-1462.

[21] van der Maaten, L. & Hinton, G. (2008) Visualizing data using t-SNE. *Journal of Machine Learning Research* **9**: 2579-2605.

[22] Campbell, W. & Karam, Z. (2009) Speaker comparison with inner product discriminant functions. In *Advances in Neural Information Processing Systems 22*, Cambridge, MA: MIT Press.

[23] Kotti, M., Moschou, V. & Kotropoulos, C. (2008) Speaker segmentation and clustering. *Signal Processing* **88**(8): 1091-1124.

[24] Raina, R., Battle, A., Lee, H., Packer, B. & Ng, A. (2007) Self-taught learning: transfer learning from unlabeled data. *Proc. ICML*, ACM press.

[25] Goldberger, J., Roweis, S., Hinton, G. & Salakhutdinov, R., (2005) Neighbourhood component analysis. In *Advances in Neural Information Processing Systems 17*, Cambridge, MA: MIT Press.

[26] Hinton, G. & Salakhutdinov, R. (2006) Reducing the dimensionality of data with neural networks. *Science* **313**: 504-507.

